# Instabilities in Eye Movement Control: A Model of Periodic Alternating Nystagmus

**Ernst R. Dow**
Center for Biophysics and
Computational Biology,
Beckman Institute
University of Illinois at Urbana-
Champaign,Urbana, IL 61801.
edow@uiuc.edu

**Thomas J. Anastasio**
Department of Molecular and Integra-
tive Physiology, Center for Biophysics
and Computational Biology,
Beckman Institute
University of Illinois at Urbana-
Champaign, Urbana, IL 61801.
tstasio@uiuc.edu

## Abstract

Nystagmus is a pattern of eye movement characterized by smooth rota-
tions of the eye in one direction and rapid rotations in the opposite di-
rection that reset eye position. Periodic alternating nystagmus (PAN) is
a form of uncontrollable nystagmus that has been described as an un-
stable but amplitude-limited oscillation. PAN has been observed previ-
ously only in subjects with vestibulo-cerebellar damage. We describe
results in which PAN can be produced in normal subjects by prolonged
rotation in darkness. We propose a new model in which the neural cir-
cuits that control eye movement are inherently unstable, but this insta-
bility is kept in check under normal circumstances by the cerebellum.
Circumstances which alter this cerebellar restraint, such as vestibulo-
cerebellar damage or plasticity due to rotation in darkness, can lead to
PAN.

## 1 INTRODUCTION

Visual perception involves not only an operating visual sensory system, but also the abil-
ity to control eye movements. The oculomotor subsystems provide eye movement con-
trol. For example, the vestibulo-ocular reflex (VOR) maintains retinal image stability by
making slow-phase eye rotations that counterbalance head rotations, making it possible to
move and see at the same time (Wilson and Melvill Jones, 1979). The VOR makes slow-
phase eye rotations that are directed opposite to head rotations. When these ongoing
slow-phase eye rotations are interrupted by fast-phase eye rotations that reset eye posi-
tion, the resulting eye movement pattern is called nystagmus. Periodic alternating nys-

tagmus (PAN) is a congenital or acquired eye movement disorder characterized by uncontrollable nystagmus that alternates direction roughly sinusoidally with a period of 200 s to 400 s (Baloh et al., 1976; Leigh et al., 1981; Furman et al., 1990). Furman and colleagues (1990) have determined that PAN in humans is caused by lesions of parts of the vestibulo-cerebellum known as the nodulus and uvula (NU). Lesions to the NU cause PAN in the dark (Waespe et al., 1985; Angelaki and Hess, 1995). NU lesions also prevent habituation (Singleton, 1967; Waespe et al, 1985; Torte et al., 1994), which is a semi-permanent decrease in the gain (eye velocity / head velocity) of the VOR response that can be brought about by prolonged low-frequency rotational stimulation in the dark. Vestibulo-cerebellectomy in habituated goldfish causes VOR dishabituation (Dow and Anastasio, 1996). Temporary inactivation of the vestibulo-cerebellum in habituated goldfish causes temporary dishabituation and can result in a temporary PAN (Dow and Anastasio, in press). Stimulation of the NU temporarily abolish the VOR response (Fernández and Fredrickson, 1964). Cerebellar influence on the VOR may be mediated by connections between the NU and vestibular nucleus neurons, which have been demonstrated in many species (Dow, 1936; 1938).

We have previously shown that intact goldfish habituate to prolonged low-frequency (0.01 Hz) rotation (Dow and Anastasio, 1996) and that rotation at higher frequencies (0.05-0.1 Hz) causes PAN (Dow and Anastasio, 1997). We also proposed a limit-cycle model of PAN in which habituation or PAN result from an increase or decrease, respectively, of the inhibition of the vestibular nuclei by the NU. This model suggested that velocity storage, which functions to increase low-frequency VOR gain above the biophysical limits of the semicircular canals (Robinson, 1977;1981), is mediated by a potentially unstable low-frequency resonance. This instability is normally kept in check by constant suppression by the NU.

## 2 METHODS

PAN was studied in intact, experimentally naïve, comet goldfish (*carassius auratus*). Each goldfish was restrained horizontally underwater with the head at the center of a cylindrical tank. Eye movements were measured using the magnetic search coil technique (Robinson, 1963). For technical details see Dow and Anastasio (1996). The tank was centered on a horizontal rotating platform. Goldfish were rotated continuously for various durations (30 min to 2 h) in darkness at various single frequencies (0.03 - 0.17 Hz). Some data have been previously reported (Dow and Anastasio, 1997). All stimuli had peak rotational velocities of 60 deg/s. Eye position and rotator (i.e. head) velocity signals were digitized for analysis. Eye position data were digitally differentiated to compute eye velocity and fast-phases were removed. Data were analyzed and simulated using MATLAB and SIMULINK (The Mathworks, Inc.).

## 3 RESULTS

Prolonged rotation in darkness at frequencies which produced some habituation in naïve goldfish (0.03-0.17 Hz) could produce a lower-frequency oscillation in slow-phase eye velocity that was superimposed on the normal VOR response (fig 1). This lower-frequency oscillation produced a periodic alternating nystagmus (PAN). When PAN occurred, it was roughly sinusoidal and varied in period, amplitude, and onset-time. Habituation could occur simultaneously with PAN (fig 1B) or habituation could completely

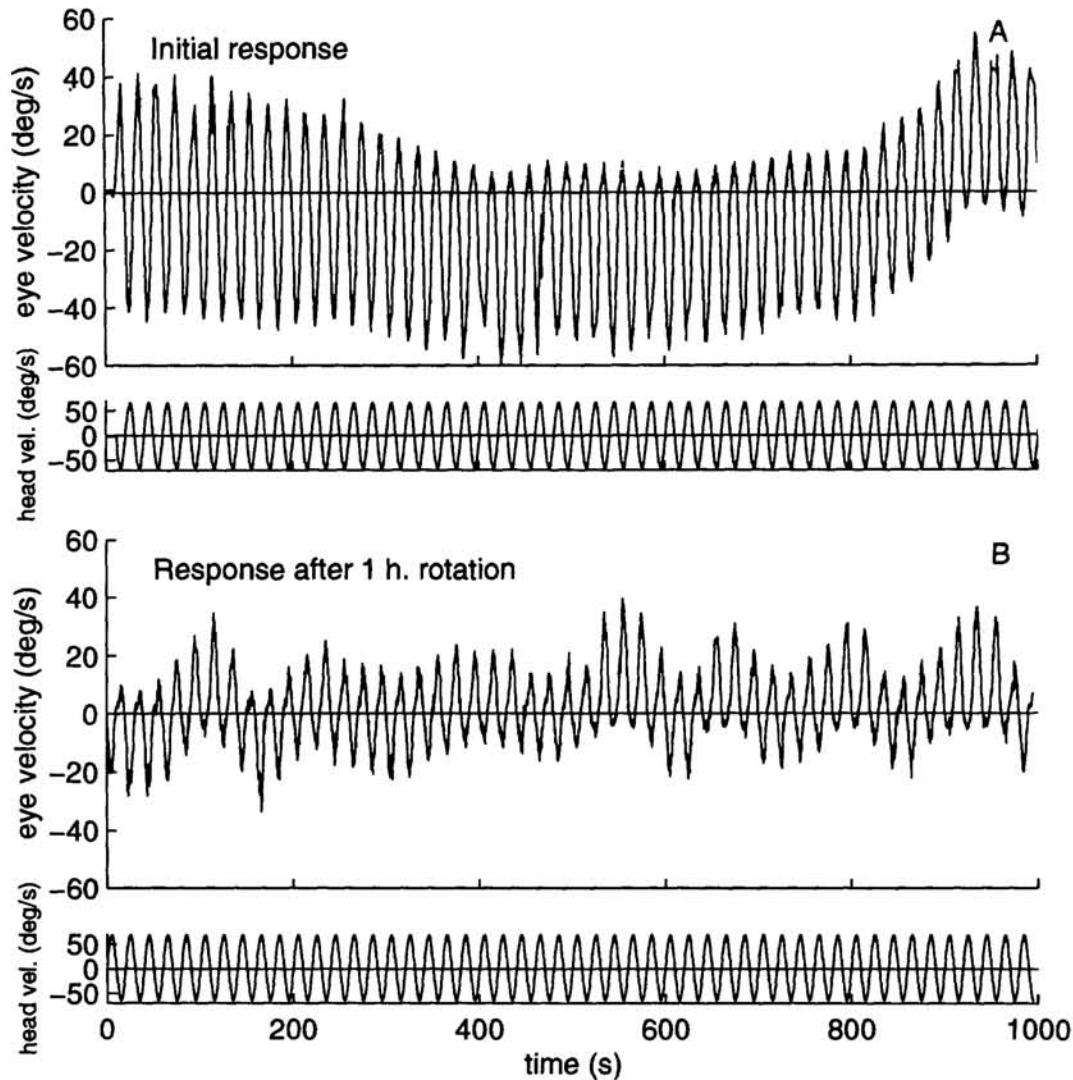

Figure 1: Initial 1000 s 0.05 Hz rotation showing PAN (**A**). Slow-phase
eye velocity shows that PAN starts almost immediately and there is a
slight reduction in VOR gain after 1000 s. Following 1 h continuous ro-
tation in the same goldfish (**B**), VOR gain has decreased.

suppress PAN (fig 4). PAN observed at lower frequencies (0.03 and 0.05 Hz) typically
decreased in amplitude as rotation continued.

Previous work has shown that PAN was most likely to occur during prolonged rotations at
frequencies between 0.05 and 0.1 Hz (Dow and Anastasio, 1997). At these frequencies,
habituation also caused a slight decrease in VOR gain (1.3 to 1.8 times, initial gain / final
gain) following 1 h of rotation. At higher frequencies, neither habituation nor PAN were
observed. At lower frequencies (0.03 Hz) PAN could occur before habituation substan-
tially reduced VOR gain (fig 4). PAN, was not observed in naïve goldfish rotated a lower
frequency (0.01 Hz) where VOR gain fell by 22 times due to habituation (Dow and An-
astasio, 1997).

# 4 MODEL

Previously, a non-linear limit cycle model was constructed by Leigh, Robinson, and Zee (1981; see also Furman, 1989) to simulate PAN in humans. This model included a velocity storage loop with saturation, and a central adaptation loop. This second order system would spontaneously oscillate, producing PAN, if the gain of the velocity storage loop was greater than 1.

We adjusted Robinson's model to simulate rotation inducible PAN and habituation in the goldfish. Input to and output from the model (fig 2) represent head and slow-phase eye velocity, respectively. The time constants of the canal ($s\tau_c/(s\tau_c+1)$) and velocity-storage ($g_s/(s\tau_s+1)$) elements were set to the value of the canal time constant as determined experimentally in goldfish ($\tau_c = \tau_s = 3$ s) (Hartman and Klinke, 1980). The time constant of the central adaptation element ($1/s\tau_a$) was 10 times longer ($\tau_a = 30$ s). The Laplace variable ($s$) is complex frequency ($s = j\omega$ where $j^2$ is $-1$ and $\omega$ is frequency in rad/s). The gain of the velocity-storage loop ($g_s$) is 1.05 while that of the central adaptation loop ($g_a$) is 1. The central adaptation loop represents in part a negative feedback loop onto vestibular nucleus neurons through inhibitory Purkinje cells of the NU. The vestibulo-cerebellum is known to modulate the gain of the VOR (Wilson and Melvill Jones, 1979). The static nonlinearity in the velocity storage loop consists of a threshold ($\pm 0.0225$) and a saturation ($\pm 1.25$). The threshold was added to model the decay in PAN following termination of rotation (Dow and Anastasio, 1997), which is not modeled here.

Increases or decreases in the absolute value of $g_a$ will cause VOR habituation or PAN, respectively. However, it was more common for VOR habituation and PAN to occur simultaneously (fig 1B). This behavior could not be reproduced with the lumped model (fig 2). It would be necessary on one hand to increase $g_a$ to decrease overall VOR gain while, on the other hand, decrease $g_a$ to produce PAN. A distributed system would address this problem, with multiple parallel pathways, each having velocity-storage and adaptive control through the NU. The idea can be illustrated using the simplest distributed system which has 2 lumped models in parallel (not shown), each having an independently adjustable $g_a$. The results from such a two parallel pathway model are shown in fig 3. In one pathway, $g_a(h)$ was increased to model habituation, and $g_a(o)$ was decreased to start oscillations. Paradoxically, although the ultimate effect of increasing $g_a(h)$ is to decrease VOR gain, the initial effect as $g_a(h)$ is increased is to increase gain. This is due to the resonant frequency of the system continuously shifting to higher frequencies and temporarily matching the frequency of rotation (see DISCUSSION). Conversely, when $g_a(o)$ is decreased, there is a temporary decrease in gain as the resonant frequency moves away from the frequency of rotation. The two results are combined after the gain is reduced by half (fig 3B).

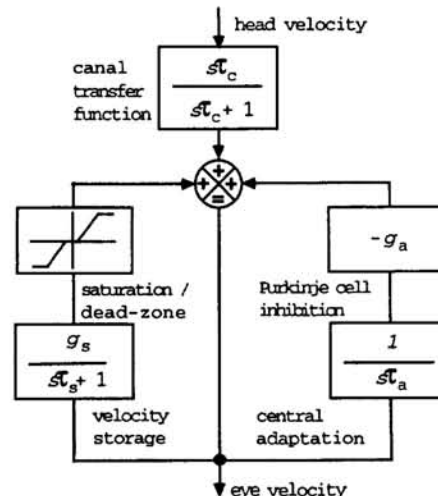

Figure 2: Model used to simulation PAN (Dow and Anastasio 1997). Used with permission.

The combined result shows a continual decrease in VOR gain with the oscillations superimposed.

## 5 DISCUSSION

If the nonlinearities (i.e. threshold and saturation) in the model are ignored, linear analysis shows that the model will be unstable when $[(1 - g_s)/\tau_s + g_d/\tau_a]$ is negative, and will oscillate with a period of $[2\pi\sqrt{(\tau_s\tau_d/g_a)}]$. With the initial parameters, the model is stable because the central adaptation loop can compensate for the unstable gain of the velocity storage loop. The natural frequency of the system, calculated from the above equation, is 0.017 Hz. This resonance, which peaks at the resonant frequency but is still pronounced at nearby frequencies, produces an enhancement of the VOR response. The hypothesis that low frequency VOR gain enhancement is produced by a potentially unstable resonance is a novel feature of our model. The natural frequency increases with increases in $g_a$ and can alter the frequency specific enhancement.

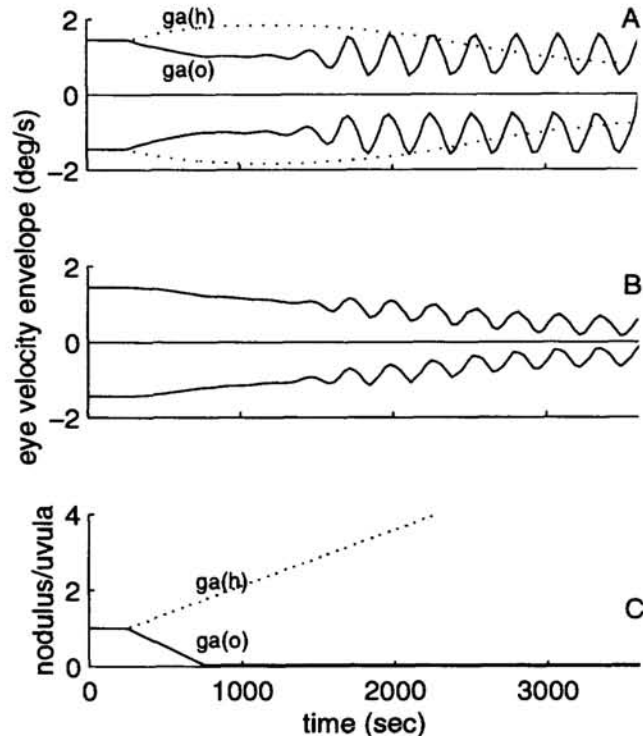

Figure 3: Model at 0.03 Hz. Two simulations with differing values of $g_a$ (A) are combined in (B) with the values of $g_a$ in (C).

Decreases in $g_a$, in addition to decreasing the natural frequency, also cause the model to become unstable. (If $g_a$ is reduced to zero, the model becomes first order and the equations are no longer valid). The ability to get either habituation or PAN by varying only one parameter suggests that habituation and PAN are a related phenomena

Through the process of habituation, prolonged low frequency rotation (0.01 Hz) in goldfish severely decreased VOR gain, often abruptly and unilaterally (Dow and Anastasio, in press). The decrease in gain due to habituation can effectively eliminate PAN at the lowest frequency at which PAN was observed (0.03 Hz) as shown in fig 4. In this example the naïve VOR responds symmetrically for the first cycle of rotation. It then becomes markedly asymmetrical, with a strong, unilateral response in one direction for ~10 cycles followed by another in the opposite direction for ~17 cycles. The VOR response abruptly habituates after that with no PAN. Complete habituation can be simulated by further increases in the value of $g_a$. in the limit-cycle model (fig 2). Unilateral habituation has been simulated previously with a bilateral network model of the VOR in which the cerebellum inhibits the vestibular nuclei unilaterally (Dow and Anastasio, in press).

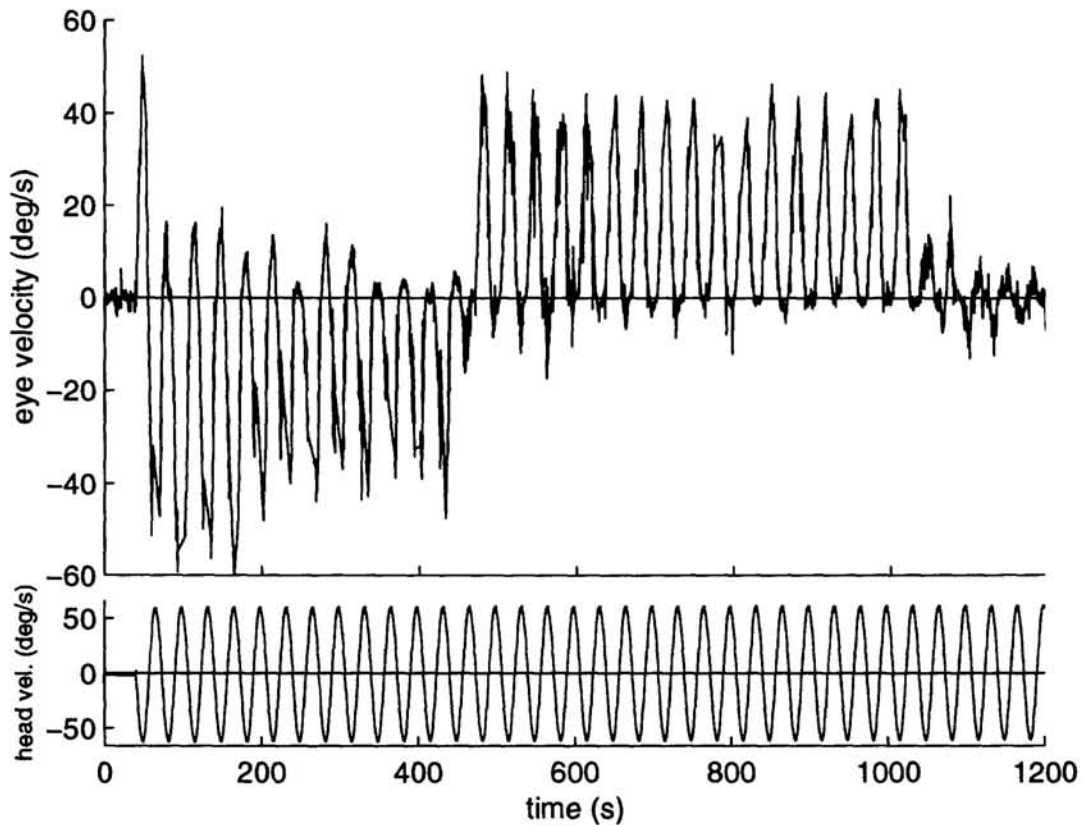

Figure 4: PAN superimposed on the VOR response to continuous rotation
at 0.03 Hz.  Upper trace, slow-phase eye velocity (fast phases removed);
lower trace, head velocity (not to scale).

The cerebellum has several circuits which could provide an increase in firing rate of some
Purkinje cells with a concurrent decrease in the firing rate of other Purkinje cells sug-
gested by the model.  There are many lateral inhibitory pathways including inhibition of
Purkinje cells to neighboring Purkinje cells (Llinás and Walton, 1990).  Therefore, if one
Purkinje cell were to increase its firing rate, this circuitry suggests that neighboring
Purkinje cells would decrease their firing rates.  Also, experimental evidence shows that
during habituation, not all vestibular nuclei gradually decrease their firing rate as might be
expected.  Kileny and colleagues (1980) recorded from vestibular nucleus neurons during
habituation.  They could divide the neurons into 3 roughly equal groups based on re-
sponse over time: continual decrease, constant followed by a decrease, and increase fol-
lowed by decrease.  The cerebellar circuitry and the single-unit recording data support
multiple, variable levels of inhibition from the NU.  How this mechanism may work is
being explored with a more biologically realistic distributed model.

## 6 CONCLUSION

Our experimental results are consistent with a multi-parallel pathway model of the VOR.
In each pathway an unstable, positive velocity-storage loop is stabilized by an inhibitory,
central adaptation loop, and their interaction produces a low-frequency resonance that
enhances the low-frequency response of the VOR.  Prolonged rotation at specific fre-
quencies could produce a decrease in central adaptation VOR gain in some pathways re-
sulting in an unstable, low-frequency oscillation resembling PAN in these pathways.  An

increase in adaptation loop gain in the other pathways would result in a decrease in VOR gain resembling habituation. The sum over the VOR pathways would show PAN and habituation occurring together. We suggest that resonance enhancement and multiple parallel (i.e. distributed) pathways are necessary to model the interrelationship between PAN and habituation.

**Acknowledgments**

The work was supported by grant MH50577 from the National Institutes of Health. We thank M. Zelaya and X. Feng for experimental assistance.

**References**

Angelaki DE and Hess BJM. *J Neurophysl* **73** 1729-1751 (1995).

Baloh RW, Honrubia V and Konrad HR. *Brain* **99** 11-26 (1976).

Dow ER and Anastasio TJ. *NeuroReport* **7** 1305-1309 (1996).

Dow ER and Anastasio TJ. NeuroReport **8** 2755-2759 (1997).

Dow ER and Anastasio TJ. *J. Computat. Neuro.* in press.

Dow RS. *J Comp Neurol* **63** 527-548 (1936).

Dow RS. *J Comp Neurol* **68** 297-305 (1938).

Fernández C and Fredrickson JM. *Acta Otolaryngol Suppl* **192** 52-62 (1964).

Furman JMR, Wall C and Pang D. *Brain* **113** 1425-1439 (1990).

Furman JMR, Hain TC and Paige GD. *Biol Cybern* **61** 255-264 (1989).

Hartmann R and Klinke R. *Pflugers Archiv* **388** 111-121 (1980).

Kileny P, Ryu JH, McCabe BF and Abbas PJ. *Acta Otolaryngol* **90** 175-183 (1980).

Leigh RJ, Robinson DA and Zee DS. *Ann NY Acad Sci* **374** 619-635 (1981).

Llinás RR and Walton KD. Cerebellum. In: Shepherd GM ed. *The Synaptic Organization of the Brain*. Oxford: Oxford University Press, 1990: 214-245.

Remmel RS. *IEEE Trans Biomed Eng* **31** 388-390 (1984).

Robinson DA. *IEEE Trans Biomed Eng* **10** 137-145 (1963).

Robinson DA. *Exp Brain Res* **30** 447-450 (1977).

Robinson DA. *Ann Rev Neurosci* **4** 463-503 (1981).

Singleton GT. *Laryngoscope* **77** 1579-1620 (1967).

Torte MP, Courjon JH, Flandrin JM, et al. *Exp Brain Res* **99** 441-454 (1994).

Waespe W, Cohen B and Raphan T. *Science* **228** 199-202 (1985).

Wilson V and Melvill Jones G. *Mammalian Vestibular Physiology*, New York: Plenum Press, 1979.